# A Kullback-Leibler Divergence Based Kernel for SVM Classification in Multimedia Applications

**Pedro J. Moreno   Purdy P. Ho**
Hewlett-Packard
Cambridge Research Laboratory
Cambridge, MA 02142, USA
{pedro.moreno,purdy.ho}@hp.com

**Nuno Vasconcelos**
UCSD ECE Department
9500 Gilman Drive, MC 0407
La Jolla, CA 92093-0407
nuno@ece.ucsd.edu

## Abstract

Over the last years significant efforts have been made to develop kernels that can be applied to sequence data such as DNA, text, speech, video and images. The Fisher Kernel and similar variants have been suggested as good ways to combine an underlying generative model in the feature space and discriminant classifiers such as SVM's. In this paper we suggest an alternative procedure to the Fisher kernel for systematically finding kernel functions that naturally handle variable length sequence data in multimedia domains. In particular for domains such as speech and images we explore the use of kernel functions that take full advantage of well known probabilistic models such as Gaussian Mixtures and single full covariance Gaussian models. We derive a kernel distance based on the Kullback-Leibler (KL) divergence between generative models. In effect our approach combines the best of both generative and discriminative methods and replaces the standard SVM kernels. We perform experiments on speaker identification/verification and image classification tasks and show that these new kernels have the best performance in speaker verification and mostly outperform the Fisher kernel based SVM's and the generative classifiers in speaker identification and image classification.

## 1   Introduction

During the last years Support Vector Machines (SVM's) [1] have become extremely successful discriminative approaches to pattern classification and regression problems. Excellent results have been reported in applying SVM's in multiple domains. However, the application of SVM's to data sets where each element has variable length remains problematic. Furthermore, for those data sets where the elements are represented by large sequences of vectors, such as speech, video or image recordings, the direct application of SVM's to the original vector space is typically unsuccessful.

While most research in the SVM community has focused on the underlying learning algorithms the study of kernels has also gained importance recently. Standard kernels such as linear, Gaussian, or polynomial do not take full advantage of the nuances of specific data sets. This has motivated plenty of research into the use of alternative kernels in the

areas of multimedia. For example, [2] applies normalization factors to polynomial kernels for speaker identification tasks. Similarly, [3] explores the use of heavy tailed Gaussian kernels in image classification tasks. These approaches in general only try to tune standard kernels (linear, polynomial, Gaussian) to the nuances of multimedia data sets.

On the other hand statistical models such as Gaussian Mixture Models (GMM) or Hidden Markov Models make strong assumptions about the data. They are simple to learn and estimate, and are well understood by the multimedia community. It is therefore attractive to explore methods that combine these models and discriminative classifiers. The Fisher kernel proposed by Jaakkola [4] effectively combines both generative and discriminative classifiers for variable length sequences. Besides its original application in genomic problems it has also been applied to multimedia domains, among others [5] applies it to audio classification with good results; [6] also tries a variation on the Fisher kernel on phonetic classification tasks.

We propose a different approach to combine both discriminative and generative methods to classification. Instead of using these standard kernels, we leverage on successful generative models used in the multimedia field. We use diagonal covariance GMM's and full covariance Gaussian models to better represent each individual audio and image object. We then use a metric derived from the symmetric Kullback-Leibler (KL) divergence to effectively compute inner products between multimedia objects.

## 2   Kernels for SVM's

Much of the flexibility and classification power of SVM's resides in the choice of kernel. Some examples are linear, polynomial degree $p$, and Gaussian. These kernel functions have two main disadvantages for multimedia signals. First they only model inner products between individual feature vectors as opposed to an ensemble of vectors which is the typical case for multimedia signals. Secondly these kernels are quite generic and do not take advantage of the statistics of the individual signals we are targeting.

The Fisher kernel approach [4] is a first attempt at solving these two issues. It assumes the existence of a generative model that explains well all possible data. For example, in the case of speech signals the generative model $p(\mathbf{x}|\boldsymbol{\theta})$ is often a Gaussian mixture. Where the $\boldsymbol{\theta}$ model parameters are priors, means, and diagonal covariance matrices. GMM's are also quite popular in the image classification and retrieval domains; [7] shows good results on image classification and retrieval using Gaussian mixtures.

For any given sequence of vectors defining a multimedia object $X = \{\mathbf{x}_1, \mathbf{x}_2, \ldots, \mathbf{x}_m\}$ and assuming that each vector in the sequence is independent and identically distributed, we can easily define the likelihood of the ensemble being generated by $p(\mathbf{x}|\boldsymbol{\theta})$ as $P(X|\boldsymbol{\theta}) = \prod_{i=1}^{m} p(\mathbf{x}_i|\boldsymbol{\theta})$. The Fisher score maps each individual sequence $\{X_1, \ldots, X_n\}$, composed of a different number of feature vectors, into a single vector in the gradient log-likelihood space.

This new feature vector, the Fisher score, is defined as

$$\mathbf{U_X} = \nabla_{\boldsymbol{\theta}} log(P(X|\boldsymbol{\theta})) \tag{1}$$

Each component of $\mathbf{U_X}$ is a derivative of the log-likelihood of the vector sequence $X$ with respect to a particular parameter of the generative model. In our case the parameters $\boldsymbol{\theta}$ of the generative model are chosen from either the prior probabilities, the mean vector or the diagonal covariance matrix of each individual Gaussian in the mixture model. For example, if we use the mean vectors as our model parameters $\boldsymbol{\theta}$, *i.e.*, for $\boldsymbol{\theta} = \boldsymbol{\mu}_k$ out of $K$ possible mixtures, then the Fisher score is

$$\nabla_{\boldsymbol{\mu}_k} log(P(X|\boldsymbol{\mu}_k)) = \sum_{i=1}^{m} P(k|\mathbf{x}_i)\boldsymbol{\Sigma}_k^{-1}(\mathbf{x}_i - \boldsymbol{\mu}_k) \quad (2)$$

where $P(k|\mathbf{x}_i)$ represents the *a posteriori* probability of mixture $k$ given the observed feature vector $\mathbf{x}_i$. Effectively we transform each multimedia object (audio or image) $X$ of variable length into a single vector $\mathbf{U_X}$ of fixed dimension.

## 3  Kullback-Leibler Divergence Based Kernels

We start with a statistical model $p(\mathbf{x}|\boldsymbol{\theta}_i)$ of the data, *i.e.*, we estimate the parameters $\boldsymbol{\theta}_i$ of a generic probability density function (PDF) for each multimedia object (utterance or image) $X_i = \{\mathbf{x}_1, \mathbf{x}_2, \ldots, \mathbf{x}_m\}$. We pick PDF's that have been shown over the years to be quite effective at modeling multimedia patterns. In particular we use diagonal Gaussian mixture models and single full covariance Gaussian models. In the first case the parameters $\boldsymbol{\theta}_i$ are priors, mean vectors, and diagonal covariance matrices while in the second case the parameters $\boldsymbol{\theta}_i$ are the mean vector and full covariance matrix.

Once the PDF $p(\mathbf{x}|\boldsymbol{\theta}_i)$ has been estimated for each training and testing multimedia object we replace the kernel computation in the original sequence space by a kernel computation in the PDF space:

$$K(X_i, X_j) \Longrightarrow K(p(\mathbf{x}|\boldsymbol{\theta}_i), p(\mathbf{x}|\boldsymbol{\theta}_j)) \quad (3)$$

To compute the PDF parameters $\boldsymbol{\theta}_i$ for a given object $X_i$ we use a maximum likelihood approach. In the case of diagonal mixture models there is no analytical solution for $\boldsymbol{\theta}_i$ and we use the Expectation Maximization algorithm. In the case of single full covariance Gaussian model there is a simple analytical solution for the mean vector and covariance matrix. Effectively we are proposing to map the input space $X_i$ to a new feature space $\boldsymbol{\theta}_i$.

Notice that if the number of vector in the $X_i$ multimedia sequence is small and there is not enough data to accurately estimate $\boldsymbol{\theta}_i$ we can use regularization methods, or even replace the maximum likelihood solution for $\boldsymbol{\theta}_i$ by a *maximum a posteriori* solution. Other solutions like starting from a generic PDF and adapting its parameters $\boldsymbol{\theta}_i$ to the current object are also possible.

The next step is to define the kernel distance in this new feature space. Because of the statistical nature of the feature space a natural choice for a distance metric is one that compares PDF's. From the standard statistical literature there are several possible choices, however, in this paper we only report our results on the symmetric Kullback-Leibler (KL) divergence

$$D(p(\mathbf{x}|\boldsymbol{\theta}_i), p(\mathbf{x}|\boldsymbol{\theta}_j)) = \int_{-\infty}^{\infty} p(\mathbf{x}|\boldsymbol{\theta}_i)\, log(\frac{p(\mathbf{x}|\boldsymbol{\theta}_i)}{p(\mathbf{x}|\boldsymbol{\theta}_j)})\, d\mathbf{x} + \int_{-\infty}^{\infty} p(\mathbf{x}|\boldsymbol{\theta}_j)\, log(\frac{p(\mathbf{x}|\boldsymbol{\theta}_j)}{p(\mathbf{x}|\boldsymbol{\theta}_i)})\, d\mathbf{x}$$

$$(4)$$

Because a matrix of kernel distances directly based on symmetric KL divergence does not satisfy the Mercer conditions, *i.e.*, it is not a positive definite matrix, we need a further step to generate a valid kernel. Among many posibilities we simply exponentiate the symmetric KL divergence, scale, and shift ($A$ and $B$ factors below) it for numerical stability reasons

$$K(X_i, X_j) \Longrightarrow K(p(\mathbf{x}|\boldsymbol{\theta}_i), p(\mathbf{x}|\boldsymbol{\theta}_j))$$
$$\Longrightarrow e^{-A\, D(p(\mathbf{x}|\boldsymbol{\theta}_i), p(\mathbf{x}|\boldsymbol{\theta}_j)) + B} \quad (5)$$

In the case of Gaussian mixture models the computation of the KL divergence is not direct. In fact there is no analytical solution to Eq. (4) and we have to resort to Monte Carlo methods or numerical approximations. In the case of single full covariance models the KL divergence has an analytical solution

$$
\begin{aligned}
D(p(\mathbf{x}|\boldsymbol{\theta}_i), p(\mathbf{x}|\boldsymbol{\theta}_j)) = tr(\boldsymbol{\Sigma}_i\,\boldsymbol{\Sigma}_j^{-1}) + tr(\boldsymbol{\Sigma}_j\,\boldsymbol{\Sigma}_i^{-1}) - \\
2\,S + tr((\boldsymbol{\Sigma}_i^{-1} + \boldsymbol{\Sigma}_j^{-1})\,(\boldsymbol{\mu}_i - \boldsymbol{\mu}_j)(\boldsymbol{\mu}_i - \boldsymbol{\mu}_j)^T)
\end{aligned}
\tag{6}
$$

where $S$ is the dimensionality of the original feature data $\mathbf{x}$. This distance is similar to the Arithmetic harmonic sphericity (AHS) distance quite popular in the speaker identification and verification research community [8].

Notice that there are significant differences between our KL divergence based kernel and the Fisher kernel method. In our approach there is no underlying generative model to represent all the data. We do not use a single PDF (even if it encodes a latent variable indicative of class membership) as a way to map the multimedia object from the original feature vector space to a gradient log-likelihood vector space. Instead each individual object (consisting of a sequence of feature vectors) is modeled by its unique PDF. This represents a more localized version of the Fisher kernel underlying generative model. Effectively the modeling power is spent where it matters most, on each of the individual objects in the training and testing sets. Interestingly, the object PDF does not have to be extremely complex. As we will show in our experimental section a single full covariance Gaussian model produces extremely good results. Also, in our approach there is not a true intermediate space unlike the gradient log-likelihood space used in the Fisher kernel. Our multimedia objects are transformed directly into PDF's.

## 4   Audio and Image Databases

We chose the 50 most frequent speakers from the HUB4-96 [9] News Broadcasting corpus and 50 speakers from the Narrowband version of the KING corpus [10] to train and test our new kernels on speaker identification and verification tasks. The HUB training set contains about 25 utterances (each 3-7 seconds long) from each speaker, resulting in 1198 utterances (or about 2 hours of speech). The HUB test set contains the rest of the utterances from these 50 speakers resulting in 15325 utterances (or about 21 hours of speech). The KING corpus is commonly used for speaker identification and verification in the speech community [11]. Its training set contains 4 utterances (each about 30 seconds long) from each speaker and the test set contains the remaining 6 from these 50 speakers. A total of 200 training utterances (about 1.67 hours of speech) and 300 test utterances (about 2.5 hours of speech) were used. Following standard practice in speech processing each utterance was transformed into a sequence of 13 dimensional Mel-Frequency Cepstral vectors. The vectors were augmented with their first and second order time derivatives resulting in a 39 dimensional feature vector. We also mean-normalized the KING utterances in order to compensate for the distortion introduced by different telephone channels. We did not do so for the HUB experiments since mean normalizing the audio would remove important speaker characteristics.

We chose the Corel database [12] to train and test all algorithms on image classification. COREL contains a variety of objects, such as landscape, vehicles, plants, and animals. To make the task more challenging we picked 8 classes of highly confusable objects: Apes, ArabianHorses, Butterflies, Dogs, Owls, PolarBears, Reptiles, and RhinosHippos. There were 100 images per class – 66 for training and 34 for testing; thus, a total of 528 training images and 272 testing images were used. All images are 353x225 pixel 24-bit RGB-color JPEGs. To extract feature vectors we followed standard practice in image processing. For

each of the 3 color channels the image was scanned by an 8x8 window shifted every 4 pixels. The 192 pixels under each window were converted into a 192-dimensional Discrete Cosine Transform (DCT) feature vector. After this only the 64 low frequency elements were used since they captured most of the image characteristics.

## 5   Experiments and Results

Our experiments trained and tested five different types of classifiers: Baseline GMM, Baseline AHS[1], SVM using Fisher kernel, and SVM using our new KL divergence based kernels.

When training and testing our new GMM/KL Divergence based kernels, a sequence of feature vectors, $\{\mathbf{x}_1, \mathbf{x}_2, \ldots, \mathbf{x}_m\}$ from each utterance or image $X$ was modeled by a single GMM of diagonal covariances. Then the KL divergences between each of these GMM's were computed according to Eq. (4) and transformed according to Eq. (5). This resulted in kernel matrices for training and testing that could be feed directly into a SVM classifier. Since all our SVM experiments are multiclass experiments we used the 1-vs-all training approach. The class with the largest positive score was designated as the winner class. For the experiments in which the object PDF was a single full covariance Gaussian we followed a similar procedure. The KL divergences between each pair of PDF's were computed according to Eq. (6) and transformed according Eq. (5). The dimensions of the resulting training and testing kernel matrices are shown in Table 1.

Table 1: *Dimensions of the training and testing kernel matrices of both new probablisitic kernels on HUB, KING, and COREL databases.*

| HUB Training | HUB Testing | KING Training | KING Testing | COREL Training | COREL Testing |
|---|---|---|---|---|---|
| 1198x1198 | 15325x1198 | 200x200 | 300x200 | 528x528 | 272x528 |

In the Fisher kernel experiments we computed the Fisher score vector $\mathbf{U_X}$ for each training and testing utterance and image with $\boldsymbol{\theta}$ parameter based on the prior probabilities of each mixture Gaussian. The underlying generative model was the same one used for the GMM classification experiments.

The task of speaker verification is different from speaker identification. We make a binary decision of whether or not an unknown utterance is spoken by the person of the claimed identity. Because we have trained SVM's using the one-vs-all approach their output can be directly used in speaker verification. To verify whether the utterance belongs to class A we just use the A-vs-all SVM output. On the other hand, the scores of the GMM and AHS classifiers cannot be used directly for verification experiments. We need to somehow combine the scores from the non claimed identities, *i.e.*, if we want to verify whether an utterance belongs to speaker A we need to compute a model for non-A speakers. This non-class model can be computed by first pooling the 49 non-class GMM's together to form a super GMM with 256x49 mixtures, (each speaker GMM has 256 mixtures). Then the score produced by this super GMM is subtracted from the score produced by the claimed speaker GMM. In the case of AHS classifiers we estimate the non-class score as the arithmetic mean of the other 49 speaker scores. To compute the miss and false positive rates we compare the

decision scores to a threshold $\Theta$. By varying $\Theta$ we can compute Detection Error Tradeoff (DET) curves as the ones shown in Fig. 1.

We compare the performance of all the 5 classifiers in speaker verification and speaker identification tasks. Table 2 shows equal-error rates (EER's) for speaker verification and accuracies of speaker identification for both speech corpora.

Table 2: *Comparison of all the classifiers used on the HUB and KING corpora. Both classification accuracy (Acc) and equal error rates (EER) are reported in percentage points.*

| Type of Classifier | HUB Acc | HUB EER | KING Acc | KING EER |
|---|---|---|---|---|
| GMM | 87.4 | 8.1 | 68.0 | 16.1 |
| AHS | 81.7 | 9.1 | 48.3 | 26.8 |
| SVM Fisher | 62.4 | 14.0 | 48.0 | 12.3 |
| SVM GMM/KL | 83.8 | 7.8 | 72.7 | 7.9 |
| SVM COV/KL | 84.7 | 7.4 | 79.7 | 6.6 |

We also compared the performance of 4 classifiers in the image classification task. Since the AHS classifier is not a effective image classifier we excluded it here. Table 3 shows the classification accuracies.

Table 3: *Comparison of the 4 classifiers used on the COREL animal subset. Classification accuracies are reported in percentage points.*

| Type of Classifier | Accuracy |
|---|---|
| GMM | 82.0 |
| SVM Fisher | 73.5 |
| SVM GMM/KL | 85.3 |
| SVM COV/KL | 80.9 |

Our results using the KL divergence based kernels in both multimedia data types are quite promising. In the case of the HUB experiments all classifiers perform similarly in both speaker verification and identification tasks with the exception of the SVM Fisher which performs significantly worse. However, For the KING database, we can see that our KL based SVM kernels outperform all other classifiers in both identification and verification tasks. Interestingly the Fisher kernel performs quite poorly too. Looking at the DET plots for both corpora we can see that on the HUB experiments the new SVM kernels perform quite well and on the KING corpora they perform much better than any other verification system.

In image classification experiments with the COREL database both KL based SVM kernels outperform the Fisher SVM; the GMM/KL kernel even outperforms the baseline GMM classifier.

# 6  Conclusion and Future Work

In this paper we have proposed a new method of combining generative models and discriminative classifiers (SVM's). Our approach is extremely simple. For every multimedia object represented by a sequence of vectors, a PDF is learned using maximum likelihood approaches. We have experimented with PDF's that are commonly used in the multimedia

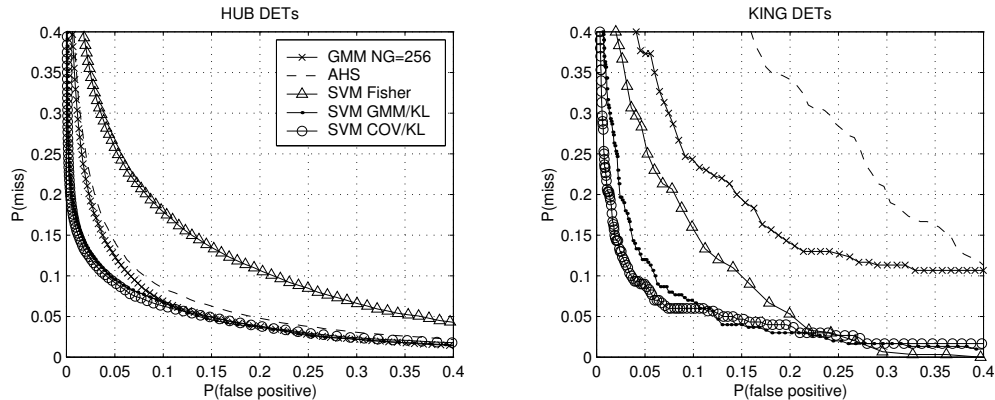

Figure 1: *Speaker verification detection error tradeoff (DET) curves for the HUB and the KING corpora, tested on all 50 speakers.*

community. However, the method is generic enough and could be used with any PDF. In the case of GMM's we use the EM algorithm to learn the model parameters $\boldsymbol{\theta}$. In the case of a single full covariance Gaussian we directly estimate its parameters. We then introduce the idea of computing kernel distances via a direct comparison of PDF's. In effect we replace the standard kernel distance on the original data $K(X_i, X_j)$ by a new kernel derived from the symmetric Kullback-Leibler (KL) divergence $K(X_i, X_j) \longrightarrow K(p(\mathbf{x}|\boldsymbol{\theta}_i), p(\mathbf{x}|\boldsymbol{\theta}_j))$. After that a kernel matrix is computed and a traditional SVM can be used.

In our experiments we have validated this new approach in speaker identification, verification, and image classification tasks by comparing its performance to Fisher kernel SVM's and other well-known classification algorithms: GMM and AHS methods. Our results show that our new method of combining generative models and SVM's always outperform the SVM Fisher kernel and the AHS methods, and it often outperforms other classification methods such as GMM's and AHS. The equal error rates are consistently better with the new kernel SVM methods too. In the case of image classification our GMM/KL divergence-based kernel has the best performance among the four classifiers while our single full covariance Gaussian distance based kernel outperforms most other classifiers and only do slightly worse than the baseline GMM. All these encouraging results show that SVM's can be improved by paying careful attention to the nature of the data being modeled. In both audio and image tasks we just take advantage of previous years of research in generative methods.

The good results obtained using a full covariance single Gaussian KL kernel also make our algorithm a very attractive alternative as opposed to the more complex methods of tuning system parameters and combining generative classifiers and discriminative methods such as the Fisher SVM. This full covariance single Gaussian KL kernel's performance is consistently good across all databases. It is especially simple and fast to compute and requires no tuning of system parameters.

We feel that this approach of combining generative classifiers via KL divergences of derived PDF's is quite generic and can possibly be applied to other domains. We plan to explore its use in other multimedia related tasks.

## Footnotes

[1]Arithmetic harmonic sphericity classifiers pull together all vectors belonging to a class and fit a single full covariance Gaussian model to the data. Similarly, a single full covariance model is fitted to each testing utterance. The similarity between the testing utterances and the class models is measured according to Eq. (6). The class with the minimum distance is chosen as the winning class.

# References

[1] Vapnik, V., *Statistical learning theory*, John Wiley and Sons, New York, 1998.

[2] Wan, V. and Campbell, W., "Support vector machines for speaker verification and identification," *IEEE Proceeding*, 2000.

[3] Chapelle, O. and Haffner, P. and Vapnik V., "Support vector machines for histogram-based image classification," *IEEE Transactions on Neural Networks*, vol. 10, no. 5, pp. 1055–1064, September 1999.

[4] Jaakkola, T., Diekhans, M. and Haussler, D., "Using the fisher kernel method to detect remote protein homologies," in *Proceedings of the Internation Conference on Intelligent Systems for Molecular Biology*, Aug. 1999.

[5] Moreno, P. J. and Rifkin, R., "Using the fisher kernel method for web audio classification," *ICASSP*, 2000.

[6] Smith N., Gales M., and Niranjan M., "Data dependent kernels in SVM classification of speech patterns," *Tech. Rep. CUED/F-INFENG/TR.387,Cambridge University Engineering Department*, 2001.

[7] Vasconcelos, N. and Lippman, A., "A unifying view of image similarity," *IEEE International Conference on Pattern Recognition*, 2000.

[8] Bimbot, F., Magrin-Chagnolleau, I. and Mathan, L., "Second-order statistical measures for text-independent speaker identification," *Speech Communication*, vol. 17, pp. 177–192, 1995.

[9] Stern, R. M., "Specification of the 1996 HUB4 Broadcast News Evaluation," in *DARPA Speech Recognition Workshop*, 1997.

[10] "The KING Speech Database," http://www.ldc.upenn.edu/Catalog/docs/LDC95S22/ kingdb.txt.

[11] Chen K., "Towards better making a decision in speaker verification," *Pattern Recognition*, , no. 36, pp. 329–246, 2003.

[12] "Corel stock photos," http://elib.cs.berleley.edu/photos/blobworld/cdlist.html.
